# Spatial distance dependent Chinese restaurant processes for image segmentation

**Soumya Ghosh**[1], **Andrei B. Ungureanu**[2], **Erik B. Sudderth**[1], and **David M. Blei**[3]

[1]Department of Computer Science, Brown University, {`sghosh,sudderth`}`@cs.brown.edu`
[2]Morgan Stanley, `andrei.b.ungureanu@gmail.com`
[3]Department of Computer Science, Princeton University, `blei@cs.princeton.edu`

## Abstract

The *distance dependent Chinese restaurant process* (ddCRP) was recently introduced to accommodate random partitions of non-exchangeable data [1]. The ddCRP clusters data in a biased way: each data point is more likely to be clustered with other data that are near it in an external sense. This paper examines the ddCRP in a spatial setting with the goal of natural image segmentation. We explore the biases of the spatial ddCRP model and propose a novel hierarchical extension better suited for producing "human-like" segmentations. We then study the sensitivity of the models to various distance and appearance hyperparameters, and provide the first rigorous comparison of nonparametric Bayesian models in the image segmentation domain. On unsupervised image segmentation, we demonstrate that similar performance to existing nonparametric Bayesian models is possible with substantially simpler models and algorithms.

## 1 Introduction

The Chinese restaurant process (CRP) is a distribution on partitions of integers [2]. When used in a mixture model, it provides an alternative representation of a Bayesian nonparametric Dirichlet process mixture—the data are clustered and the number of clusters is determined via the posterior distribution. CRP mixtures assume that the data are exchangeable, i.e., their order does not affect the distribution of cluster structure. This can provide computational advantages and simplify approximate inference, but is often an unrealistic assumption.

The *distance dependent Chinese restaurant process* (ddCRP) was recently introduced to model random partitions of non-exchangeable data [1]. The ddCRP clusters data in a biased way: each data point is more likely to be clustered with other data that are near it in an external sense. For example, when clustering time series data, points that closer in time are more likely to be grouped together. Previous work [1] developed the ddCRP mixture in general, and derived posterior inference algorithms based on Gibbs sampling [3]. While they studied the ddCRP in time-series and sequential settings, ddCRP models can be used with any type of distance and external covariates. Recently, other researchers [4] have also used the ddCRP in non-temporal settings.

In this paper, we study the ddCRP in a spatial setting. We use a spatial distance function between pixels in natural images and cluster them to provide an unsupervised segmentation. The spatial distance encourages the discovery of connected segments. We also develop a region-based hierarchical generalization, the rddCRP. Analogous to the hierarchical Dirichlet process (HDP) [5], the rddCRP clusters groups of data, where cluster components are shared across groups. Unlike the HDP, however, the rddCRP allows within-group clusterings to depend on external distance measurements.

To demonstrate the power of this approach, we develop posterior inference algorithms for segmenting images with ddCRP and rddCRP mixtures. Image segmentation is an extensively studied area,

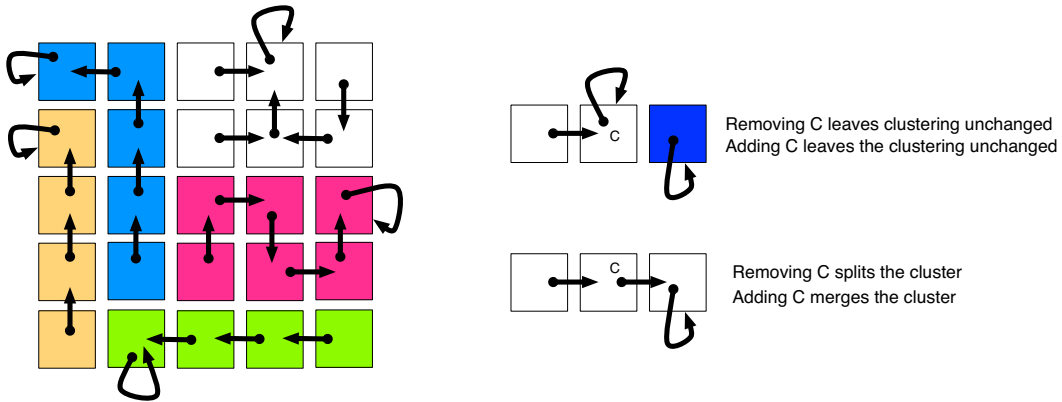

Figure 1: *Left:* An illustration of the relationship between the customer assignment representation and the table assignment representation. Each square is a data point (a pixel or superpixel) and each arrow is a customer assignment. Here, the distance window is of length 1. The corresponding table assignments, i.e., the clustering of these data, is shown by the color of the data points. *Right:* Intuitions behind the two cases considered by the Gibbs sampler. Consider the link from node $C$. When removed, it may leave the clustering unchanged or split a cluster. When added, it may leave the clustering unchanged or merge two clusters.

which we will not attempt to survey here. Influential existing methods include approaches based on kernel density estimation [6], Markov random fields [3, 7], and the normalized cut spectral clustering algorithm [8, 9]. A recurring difficulty encountered by traditional methods is the need to determine an appropriate segment resolution for each image; even among images of similar scene types, the number of observed objects can vary widely. This has usually been dealt via heuristics with poorly understood biases, or by simplifying the problem (e.g., partially specifying each image's segmentation via manual user input [7]).

Recently, several promising segmentation algorithms have been proposed based on nonparametric Bayesian methods [10, 11, 12]. In particular, an approach which couples Pitman-Yor mixture models [13] via thresholded Gaussian processes [14] has lead to very promising initial results [10], and provides a baseline for our later experiments. Expanding on the experiments in [10], we analyze 800 images of different natural scene types, and show that the comparatively simpler ddCRP-based algorithms perform similarly to this work. Moreover, unlike previous nonparametric Bayesian approaches, the structure of the ddCRP allows spatial connectivity of the inferred segments to (optionally) be enforced. In some applications, this is a known property of all reasonable segmentations.

Our results demonstrate the practical utility of spatial ddCRP and hierarchical rddCRP models. We also provide the first rigorous comparison of nonparametric Bayesian image segmentation models.

## 2 Image Segmentation with Distance Dependent CRPs

Our goal is to develop a probabilistic method to segment images of complex scenes. Image segmentation is the problem of partitioning an image into self-similar groups of adjacent pixels. Segmentation is an important step towards other tasks in image understanding, such as object recognition, detection, or tracking. We model images as observed collections of "superpixels" [15], which are small blocks of spatially adjacent pixels. Our goal is to associate the features $x_i$ in the $i^{th}$ superpixel with some cluster $z_i$; these clusters form the segments of that image.

Image segmentation is thus a special kind of clustering problem where the desired solution has two properties. First, we hope to find contiguous regions of the image assigned to the same cluster. Due to physical processes such as occlusion, it may be appropriate to find segments that contain two or three contiguous image regions, but we do not want a cluster that is scattered across individual image pixels. Traditional clustering algorithms, such as $k$-means or probabilistic mixture models, do not account for external information such as pixel location and are not biased towards contiguous regions. Image locations have been heuristically incorporated into Gaussian mixture models by concatenating positions with appearance features in a vector [16], but the resulting bias towards elliptical regions often produces segmentation artifacts. Second, we would like a solution that deter-

mines the number of clusters from the image. Image segmentation algorithms are typically applied to collections of images of widely varying scenes, which are likely to require different numbers of segments. Except in certain restricted domains such as medical image analysis, it is not practical to use an algorithm that requires knowing the number of segments in advance.

In the following sections, we develop a Bayesian algorithm for image segmentation based on the distance dependent Chinese restaurant process (ddCRP) mixture model [1]. Our algorithm finds spatially contiguous segments in the image and determines an image-specific number of segments from the observed data.

## 2.1 Chinese restaurant process mixtures

The ddCRP mixture is an extension of the traditional Chinese restaurant process (CRP) mixture. CRP mixtures provide a clustering method that determines the number of clusters from the data—they are an alternative formulation of the Dirichlet process mixture model. The assumed generative process is described by imagining a restaurant with an infinite number of tables, each of which is endowed with a parameter for some family of data generating distributions (in our experiments, Dirichlet). Customers enter the restaurant in sequence and sit at a randomly chosen table. They sit at the previously occupied tables with probability proportional to how many customers are already sitting at each; they sit at an unoccupied table with probability proportional to a scaling parameter. After the customers have entered the restaurant, the "seating plan" provides a clustering. Finally, each customer draws an observation from a distribution determined by the parameter at the assigned table.

Conditioned on observed data, the CRP mixture provides a posterior distribution over table assignments and the parameters attached to those tables. It is a distribution over clusterings, where the number of clusters is determined by the data. Though described sequentially, the CRP mixture is an exchangeable model: the posterior distribution over partitions does not depend on the ordering of the observed data.

Theoretically, exchangeability is necessary to make the connection between CRP mixtures and Dirichlet process mixtures. Practically, exchangeability provides efficient Gibbs sampling algorithms for posterior inference. However, exchangeability is not an appropriate assumption in image segmentation problems—the locations of the image pixels are critical to providing contiguous segmentations.

## 2.2 Distance dependent CRPs

The distance dependent Chinese Restaurant Process (ddCRP) is a generalization of the Chinese restaurant process that allows for a non-exchangeable distribution on partitions [1]. Rather than represent a partition by customers assigned to tables, the ddCRP models customers linking to other customers. The seating plan is a byproduct of these links—two customers are sitting at the same table if one can reach the other by traversing the customer assignments. As in the CRP, tables are endowed with data generating parameters. Once the partition is determined, the observed data for each customer are generated by the per-table parameters.

As illustrated in Figure 1, the generative process is described in terms of customer assignments $c_i$ (as opposed to partition assignments or tables, $z_i$). The distribution of customer assignments is

$$p\left(c_i = j \mid D, f, \alpha\right) \propto \begin{cases} f(d_{ij}) & j \neq i, \\ \alpha & j = i. \end{cases} \tag{1}$$

Here $d_{ij}$ is a distance between data points $i$ and $j$ and $f(d)$ is called the decay function. The decay function mediates how the distance between two data points affects their probability of connecting to each other, i.e., their probability of belonging to the same cluster.

Details of the ddCRP are found in [1]. We note that the traditional CRP is an instance of a ddCRP. However, in general, the ddCRP does not correspond to a model based on a random measure, like the Dirichlet process. The ddCRP is appropriate for image segmentation because it can naturally account for the spatial structure of the superpixels through its distance function. We use a spatial distance between pixels to enforce a bias towards contiguous clusters. Though the ddCRP has been previously described in general, only time-based distances are studied in [1].

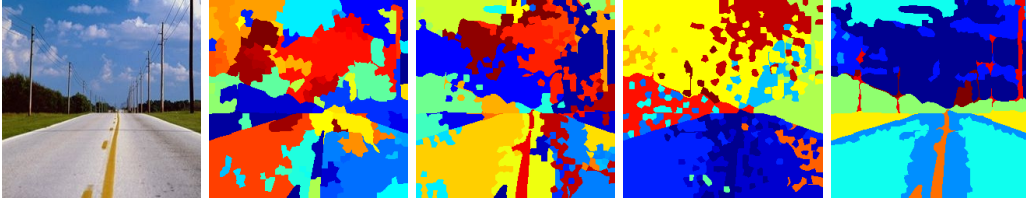

Figure 2: Comparison of distance-dependent segmentation priors. From left to right, we show segmentations produced by the ddCRP with $a = 1$, the ddCRP with $a = 2$, the ddCRP with $a = 5$, and the rddCRP with $a = 1$.

Restaurants represent images, tables represent segment assignment, and customers represent super-pixels. The distance between superpixels is modeled as the number of hops required to reach one superpixel from another, with hops being allowed only amongst spatially neighboring superpixels. A "window" decay function of width $a$, $f(d) = \mathbb{1}[d \leq a]$, determines link probabilities. If $a = 1$, superpixels can only directly connect to adjacent superpixels. Note this does not explicitly restrict the size of segments, because any pair of pixels for which one is *reachable* from the other (i.e., in the same connected component of the customer assignment graph) are in the same image segment. For this special case segments are *guaranteed* with probability one to form spatially connected subsets of the image, a property not enforced by other Bayesian nonparametric models [10, 11, 12].

The full generative process for the observed features $x_{1:N}$ within a $N$-superpixel image is as follows:

1. For each table, sample parameters $\phi_k \sim G_0$.
2. For each customer, sample a customer assignment $c_i \sim \mathrm{ddCRP}(\alpha, f, D)$. This indirectly determines the cluster assignments $z_{1:N}$, and thus the segmentation.
3. For each superpixel, independently sample observed data $x_i \sim P(\cdot \,|\, \phi_{z_i})$.

The customer assignments are sampled using the spatial distance between pixels. The partition structure, derived from the customer assignments, is used to sample the observed image features. Given an image, the posterior distribution of the customer assignments induces a posterior over the cluster structure; this provides the segmentation. See Figure 1 for an illustration of the customer assignments and their derived table assignments in a segmentation setting.

As in [10], the data generating distribution for the observed features studied in Section 4 is multinomial, with separate distributions for color and texture. We place conjugate Dirichlet priors on these cluster parameters.

### 2.3 Region-based hierarchical distance dependent CRPs

The ddCRP model, when applied to an image with window size $a = 1$, produces a collection of contiguous patches (tables) homogeneous in color and texture features (Figure 2). While such segmentations are useful for various applications [16], they do not reflect the statistics of manual human segmentations, which contain larger regions [17]. We could bias our model to produce such regions by either increasing the window size $a$, or by introducing a hierarchy wherein the produced patches are grouped into a small number of regions. This region level model has each patch (table) associated with a region $k$ from a set of potentially infinite regions. Each region in turn is associated with an appearance model $\phi_k$. The corresponding generative process is described as follows:

1. For each customer, sample customer assignments $c_i \sim \mathrm{ddCRP}(\alpha, f, D)$. This determines the table assignments $t_{1:N}$.
2. For each table $t$, sample region assignments $k_t \sim \mathrm{CRP}(\gamma)$.
3. For each region, sample parameters $\phi_k \sim G_0$.
4. For each superpixel, independently sample observed data $x_i \sim P(\cdot \,|\, \phi_{z_i})$, where $z_i = k_{t_i}$.

Note that this region level rddCRP model is a direct extension of the Chinese restaurant franchise (CRF) representation of the HDP [5], with the image partition being drawn from a ddCRP instead

of a CRP. In contrast to prior applications of the HDP, our region parameters are not shared amongst images, although it would be simple to generalize to this case. Figure 3 plots samples from the rddCRP and ddCRP priors with increasing $a$. The rddCRP produces larger partitions than the ddCRP with $a = 1$, while avoiding the noisy boundaries produced by a ddCRP with large $a$ (see Figure 2).

## 3 Inference with Gibbs Sampling

A segmentation of an observed image is found by posterior inference. The problem is to compute the conditional distribution of the latent variables—the customer assignments $c_{1:N}$—conditioned on the observed image features $x_{1:N}$, the scaling parameter $\alpha$, the distances between pixels $D$, the window size $a$, and the base distribution hyperparameter $\lambda$:

$$p(c_{1:N} \,|\, x_{1:N}, \alpha, d, a, \lambda) = \frac{\left(\prod_{i=1}^{N} p(c_i \,|\, D, a, \alpha)\right) p(x_{1:N} \,|\, z(c_{1:N}), \lambda)}{\sum_{c_{1:N}} \left(\prod_{i=1}^{N} p(c_i \,|\, D, a, \alpha)\right) p(x_{1:N} \,|\, z(c_{1:N}), \lambda)} \qquad (2)$$

where $z(c_{1:N})$ is the cluster representation that is derived from the customer representation $c_{1:N}$. Notice again that the prior term uses the customer representation to take into account distances between data points; the likelihood term uses the cluster representation.

The posterior in Equation (2) is not tractable to directly evaluate, due to the combinatorial sum in the denominator. We instead use Gibbs sampling [3], a simple form of Monte Carlo Markov chain (MCMC) inference [18]. We define the Markov chain by iteratively sampling each latent variable $c_i$ conditioned on the others and the observations,

$$p(c_i \,|\, c_{-i}, x_{1:N}, D, \alpha, \lambda) \propto p(c_i \,|\, D, \alpha) p(x_{1:N} \,|\, z(c_{1:N}), \lambda). \qquad (3)$$

The prior term is given in Equation (1). We can decompose the likelihood term as follows:

$$p(x_{1:N} \,|\, z(c_{1:N}), \lambda) = \prod_{k=1}^{K(c_{1:N})} p(x_{z(c_{1:N})=k} \,|\, z(c_{1:N}), \lambda). \qquad (4)$$

We have introduced notation to more easily move from the customer representation—the primary latent variables of our model—and the cluster representation. Let $K(c_{1:N})$ denote the number of unique clusters in the customer assignments, $z(c_{1:N})$ the cluster assignments derived from the customer assignments, and $x_{z(c_{1:N})=k}$ the collection of observations assigned to cluster $k$. We assume that the cluster parameters $\phi_k$ have been analytically marginalized. This is possible when the base distribution $G_0$ is conjugate to the data generating distribution, e.g. Dirichlet to multinomial.

Sampling from Equation (3) happens in two stages. First, we remove the customer link $c_i$ from the current configuration. Then, we consider the prior probability of each possible value of $c_i$ and how it changes the likelihood term, by moving from $p(x_{1:N} \,|\, z(c_{-i}), \lambda)$ to $p(x_{1:N} \,|\, z(c_{1:N}), \lambda)$.

In the first stage, removing $c_i$ either leaves the cluster structure intact, i.e., $z(c_{1:N}^{\text{old}}) = z(c_{-i})$, or splits the cluster assigned to data point $i$ into two clusters. In the second stage, randomly reassigning $c_i$ either leaves the cluster structure intact, i.e., $z(c_{-i}) = z(c_{1:N})$, or joins the cluster assigned to data point $i$ to another. See Figure 1 for an illustration of these cases. Via these moves, the sampler explores the space of possible segmentations.

Let $\ell$ and $m$ be the indices of the tables that are joined to index $k$. We first remove $c_i$, possibly splitting a cluster. Then we sample from

$$p(c_i \,|\, c_{-i}, x_{1:N}, D, \alpha, \lambda) \propto \begin{cases} p(c_i \,|\, D, \alpha) \Gamma(x, z, \lambda) & \text{if } c_i \text{ joins } \ell \text{ and } m; \\ p(c_i \,|\, D, \alpha) & \text{otherwise,} \end{cases} \qquad (5)$$

where

$$\Gamma(x, z, \lambda) = \frac{p(x_{z(c_{1:N})=k} \,|\, \lambda)}{p(x_{z(c_{1:N})=\ell} \,|\, \lambda) p(x_{z(c_{1:N})=m} \,|\, \lambda)}. \qquad (6)$$

This defines a Markov chain whose stationary distribution is the posterior of the spatial ddCRP defined in Section 2. Though our presentation is slightly different, this algorithm is equivalent to the one developed for ddCRP mixtures in [1].

In the rddCRP, the algorithm for sampling the customer indicators is nearly the same, but with two differences. First, when $c_i$ is removed, it may spawn a new cluster. In that case, the region identity of the new table must be sampled from the region level CRP. Second, the likelihood term in Equation (4) depends only on the superpixels in the image assigned to the segment in question. In the rddCRP, it also depends on other superpixels assigned to segments that are assigned to the same region. Finally, the rddCRP also requires resampling of region assignments as follows:

$$p(k_t = \ell \,|\, k_{-t}, x_{1:N}, t(c_{1:N}), \gamma, \lambda) \propto \left\{ \begin{array}{ll} m_\ell^{-t} p(x_t \,|\, x_{-t}, \lambda) & \text{if } \ell \text{ is used;} \\ \gamma p(x_t \,|\, \lambda) & \text{if } \ell \text{ is new.} \end{array} \right. \quad (7)$$

Here, $x_t$ is the set of customers sitting at table $t$, $x_{-t}$ is the set of all customers associated with region $\ell$ excluding $x_t$, and $m_\ell^{-t}$ is the number of tables associated with region $\ell$ excluding $x_t$.

## 4 Empirical Results

We compare the performance of the ddCRP to manual segmentations of images drawn from eight natural scene categories [19]. Non-expert users segmented each image into polygonal shapes, and labeled them as distinct objects. The collection, which is available from LabelMe [17], contains a total of 2,688 images.[1] We randomly select 100 images from each category. This image collection has been previously used to analyze an image segmentation method based on spatially dependent Pitman-Yor (PY) processes [10], and we compare both methods using an identical feature set. Each image is first divided into approximately 1000 superpixels [15, 20][2] using the normalized cut algorithm [9].[3] We describe the texture of each superpixel via a local texton histogram [21], using band-pass filter responses quantized to 128 bins. A 120-bin HSV color histogram is also computed. Each superpixel $i$ is summarized via these histograms $x_i$.

Our goal is to make a controlled comparison to alternative nonparametric Bayesian methods on a challenging task. Performance is assessed via agreement with held out human segmentations, via the Rand index [22]. We also present segmentation results for qualitative evaluation in Figures 3 and 4 .

### 4.1 Sensitivity to Hyperparameters

Our models are governed by the CRP concentration parameters $\gamma$ and $\alpha$, the appearance base measure hyperparameter $\lambda = (\lambda_0, ...\lambda_0)$, and the window size $a$. Empirically, $\gamma$ has little impact on the segmentation results, due to the high-dimensional and informative image features; all our experiments set $\gamma = 1$. $\alpha$ and $\lambda_0$ induce opposing biases: a small $\alpha$ encourages larger segments, while a large $\lambda_0$ encourages larger segments. We found $\alpha = 10^{-8}$ and $\lambda_0 = 20$ to work well.

The most influential prior parameter is the window size $a$, the effect of which is visualized in Figure 3. For the ddCRP model, setting $a = 1$ (*ddCRP1*) produces a set of small but contiguous segments. Increasing to $a = 2$ (*ddCRP2*) results in fewer segments, but the produced segments are typically spatially fragmented. This phenomenon is further exacerbated with larger values of $a$. The rddCRP model groups segments produced by a ddCRP. Because it is hard to recover meaningful partitions if these initial segments are poor, the rddCRP performs best when $a = 1$.

### 4.2 Image Segmentation Performance

We now quantitatively measure the performance of our models. The ddCRP and the rddCRP samplers were run for 100 and 500 iterations, respectively. Both samplers displayed rapid mixing and often stabilized withing the first 50 iterations. Note that similar rapid mixing has been observed in other applications of the ddCRP [1].

We also compare to two previous models [10]: a PY mixture model with no spatial dependence (*pybof20*), and a PY mixture with spatial coupling induced via thresholded Gaussian processes (*pydist20*). To control the comparison as much as possible, the PY models are tested with identical features and base measure $\beta$, and other hyperparameters as in [10]. We also compare to the non-spatial PY with $\lambda_0 = 1$, the best bag-of-feature model in our experiments (*pybof*). We employ

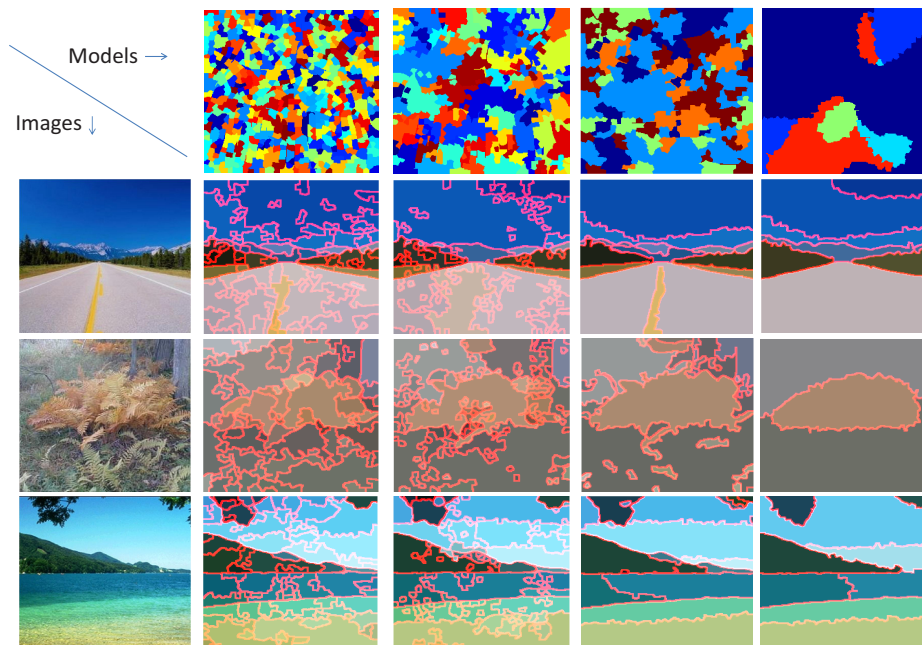

Figure 3: Segmentations produced by various Bayesian nonparametric methods. From left to right, the columns display natural images, segmentations for the ddCRP with $a = 1$, the ddCRP with $a = 2$, the rddCRP with $a = 1$, and thresholded Gaussian processes (pydist20) [10]. The top row displays partitions sampled from the corresponding priors, which have 130, 54, 5, and 6 clusters, respectively.

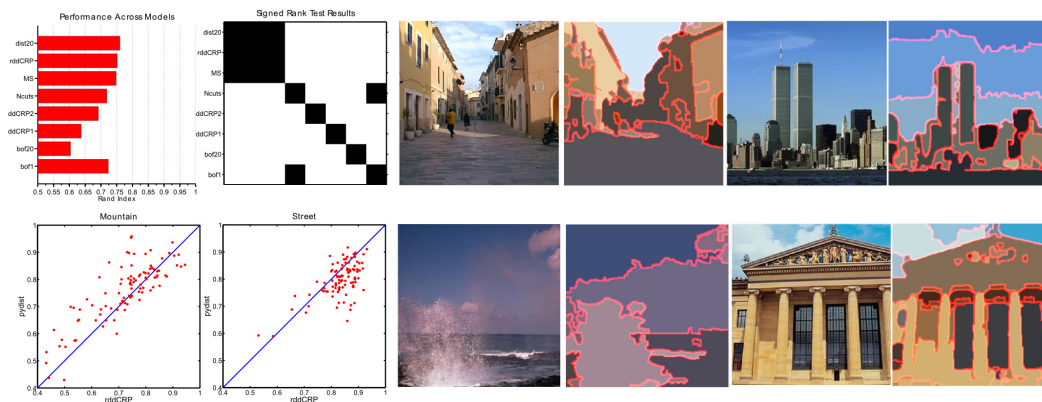

Figure 4: *Top left:* Average segmentation performance on the database of natural scenes, as measured by the Rand index (larger is better), and those pairs of methods for which a Wilcoxon's signed rank test indicates comparable performance with 95% confidence. In the binary image, dark pixels indicate pairs that are statistically indistinguishable. Note that the rddCRP, spatial PY, and mean shift methods are statistically indistinguishable, and significantly better than all others. *Bottom left:* Scatter plots comparing the pydist20 and rddCRP methods in the *Mountain* and *Street* scene categories. *Right:* Example segmentations produced by the rddCRP.

non-hierarchical versions of the PY models, so that each image is analyzed independently, and perform inference via the previously developed mean field variational method. Finally, from the vision literature we also compare to the normalized cuts (*Ncuts*) [8] and mean shift (*MS*) [6] segmentation algorithms.[4]

Quantitative performance is summarized in Figure 4. The rddCRP outscores both versions of the ddCRP model, in terms of Rand index. Nevertheless, the patchy ddCRP1 segmentations are interesting for applications where segmentation is an intermediate step rather than the final goal. The bag of features model with $\lambda_0 = 20$ performs poorly; with optimized $\lambda_0 = 1$ it is better, but still inferior to the best spatial models.

In general, the spatial PY and rddCRP perform similarly. The scatter plots in Fig. 4, which show Rand indexes for each image from the mountain and street categories, provide insights into when one model outperforms the other. For the street images rddCRP is better, while for images containing mountains spatial PY is superior. In general, street scenes contain more objects, many of which are small, and thus disfavored by the smooth Gaussian processes underlying the PY model. To most fairly compare priors, we have tested a version of the spatial PY model employing a covariance function that depends only on spatial distance. Further performance improvements were demonstrated in [10] via a conditionally specified covariance, which depends on detected image boundaries. Similar conditional specification of the ddCRP distance function is a promising direction for future research.

Finally, we note that the ddCRP (and rddCRP) models proposed here are far simpler than the spatial PY model, both in terms of model specification and inference. The ddCRP models only require pairwise superpixel distances to be specified, as opposed to the positive definite covariance function required by the spatial PY model. Furthermore, the PY model's usage of thresholded Gaussian processes leads to a complex likelihood function, for which inference is a significant challenge. In contrast, ddCRP inference is carried out through a straightforward sampling algorithm,[5] and thus may provide a simpler foundation for building rich models of visual scenes.

## 5   Discussion

We have studied the properties of spatial distance dependent Chinese restaurant processes, and applied them to the problem of image segmentation. We showed that the spatial ddCRP model is particularly well suited for segmenting an image into a collection of contiguous patches. Unlike previous Bayesian nonparametric models, it can produce segmentations with guaranteed spatial connectivity. To go from patches to coarser, human-like segmentations, we developed a hierarchical region-based ddCRP. This hierarchical model achieves performance similar to state-of-the-art nonparametric Bayesian segmentation algorithms, using a simpler model and a substantially simpler inference algorithm.

## Footnotes

[1] *http://labelme.csail.mit.edu/browseLabelMe/*

[2] *http://www.cs.sfu.ca/˜mori/*

[3] *http://www.eecs.berkeley.edu/Research/Projects/CS/vision/*

[4]We used the EDISON implementation of mean shift. The parameters of mean shift and normalized cuts were tuned by performing a grid search over a training set containing 25 images from each of the 8 categories. For normalized cuts the optimal number of segments was determined to be 5. For mean shift we held the spatial

## References

[1] D. M. Blei and P. I. Frazier. Distant dependent chinese restaurant processes. *Journal of Machine Learning Research*, 12:2461–2488, August 2011.

[2] J. Pitman. *Combinatorial Stochastic Processes*. Lecture Notes for St. Flour Summer School. Springer-Verlag, New York, NY, 2002.

[3] S. Geman and D. Geman. Stochastic relaxation, Gibbs distributions, and the Bayesian restoration of images. *IEEE Transactions on pattern analysis and machine intelligence*, 6(6):721–741, November 1984.

[4] Richard Socher, Andrew Maas, and Christopher D. Manning. Spectral chinese restaurant processes: Nonparametric clustering based on similarities. In *Fourteenth International Conference on Artificial Intelligence and Statistics (AISTATS)*, 2011.

[5] Y. W. Teh, M. I. Jordan, M. J. Beal, and D. M. Blei. Hierarchical Dirichlet processes. *Journal of American Statistical Association*, 25(2):1566 – 1581, 2006.

[6] D. Comaniciu and P. Meer. Mean shift: A robust approach toward feature space analysis. *IEEE Transactions on pattern analysis and machine intelligence*, pages 603–619, 2002.

bandwidth constant at 7, and found optimal values of feature bandwidth and minimum region size to be 25 and 4000 pixels, respectively.

[5]In our Matlab implementations, the core ddCRP code was less than half as long as the corresponding PY code. For the ddCRP, the computation time was 1 minute per iteration, and convergence typically happened after only a few iterations. The PY code, which is based on variational approximations, took 12 minutes per image.

[7] C. Rother, V. Kolmogorov, and A. Blake. Grabcut: Interactive foreground extraction using iterated graph cuts. In *ACM Transactions on Graphics (TOG)*, volume 23, pages 309–314, 2004.

[8] J. Shi and J. Malik. Normalized cuts and image segmentation. *IEEE Trans. PAMI*, 22(8):888–905, 2000.

[9] C. Fowlkes, D. Martin, and J. Malik. Learning affinity functions for image segmentation: Combining patch-based and gradient-based approaches. *CVPR*, 2:54–61, 2003.

[10] E. B. Sudderth and M. I. Jordan. Shared segmentation of natural scenes using dependent pitman-yor processes. *NIPS 22*, 2008.

[11] P. Orbanz and J. M. Buhmann. Smooth image segmentation by nonparametric Bayesian inference. In *ECCV*, volume 1, pages 444–457, 2006.

[12] Lan Du, Lu Ren, David Dunson, and Lawrence Carin. A bayesian model for simultaneous image clustering, annotation and object segmentation. In *NIPS 22*, pages 486–494. 2009.

[13] J. Pitman and M. Yor. The two-parameter Poisson–Dirichlet distribution derived from a stable subordinator. *Annals of Probability*, 25(2):855–900, 1997.

[14] J. A. Duan, M. Guindani, and A. E. Gelfand. Generalized spatial Dirichlet process models. *Biometrika*, 94(4):809–825, 2007.

[15] X. Ren and J. Malik. Learning a classification model for segmentation. *ICCV*, 2003.

[16] C. Carson, S. Belongie, H. Greenspan, and J. Malik. Blobworld: Image segmentation using expectation-maximization and its application to image querying. *PAMI*, 24(8):1026–1038, August 2002.

[17] B. C. Russell, A. Torralba, K. P. Murphy, and W. T. Freeman. Labelme: A database web-based tool for image annotation. *IJCV*, 77:157–173, 2008.

[18] C. Robert and G. Casella. *Monte Carlo Statistical Methods*. Springer Texts in Statistics. Springer-Verlag, New York, NY, 2004.

[19] A. Oliva and A. Torralba. Modeling the shape of the scene: A holistic representation of the spatial envelope. *IJCV*, 42(3):145 – 175, 2001.

[20] G. Mori. Guiding model search using segmentation. *ICCV*, 2005.

[21] D. R. Martin, C.C. Fowlkes, and J. Malik. Learning to detect natural image boundaries using local brightness, color, and texture cues. *IEEE Trans. PAMI*, 26(5):530–549, 2004.

[22] W.M. Rand. Objective criteria for the evaluation of clustering methods. *Journal of the American Statistical Association*, pages 846–850, 1971.

